# Learning from user feedback in image retrieval systems

**Nuno Vasconcelos**          **Andrew Lippman**
MIT Media Laboratory, 20 Ames St, E15-354, Cambridge, MA 02139,
{nuno,lip}@media.mit.edu,          http://www.media.mit.edu/~nuno

## Abstract

We formulate the problem of retrieving images from visual databases as a problem of Bayesian inference. This leads to natural and effective solutions for two of the most challenging issues in the design of a retrieval system: providing support for region-based queries without requiring prior image segmentation, and accounting for user-feedback during a retrieval session. We present a new learning algorithm that relies on belief propagation to account for both positive and negative examples of the user's interests.

## 1 Introduction

Due to the large amounts of imagery that can now be accessed and managed via computers, the problem of content-based image retrieval (CBIR) has recently attracted significant interest among the vision community [1, 2, 5]. Unlike most traditional vision applications, very few assumptions about the content of the images to be analyzed are allowable in the context of CBIR. This implies that the space of valid image representations is restricted to those of a generic nature (and typically of low-level) and consequently the image understanding problem becomes even more complex. On the other hand, CBIR systems have access to feedback from their users that can be exploited to simplify the task of finding the desired images. There are, therefore, two fundamental problems to be addressed. First, the design of the image representation itself and, second, the design of learning mechanisms to facilitate the interaction. The two problems cannot, however, be solved in isolation as the careless selection of the representation will make learning more difficult and vice-versa.

The impact of a poor image representation on the difficulty of the learning problem is visible in CBIR systems that rely on holistic metrics of image similarity, forcing user-feedback to be relative to entire images. In response to a query, the CBIR system suggests a few images and the user rates those images according to how well they satisfy the goals of the search. Because each image usually contains several different objects or visual concepts, this rating is both difficult and inefficient. How can the user rate an image that contains the concept of interest, but in which this concept only occupies 30% of the field of view, the remaining 70% being filled with completely unrelated stuff? And how many example images will the CBIR system have to see, in order to figure out what the concept of interest is?

A much better interaction paradigm is to let the user explicitly select the regions of the image that are relevant to the search, i.e. user-feedback at the region level. However, region-based feedback requires sophisticated image representations. The problem is that the most obvious choice, object-based representations, is difficult to implement because it is still too hard to segment arbitrary images in a meaningful way. We have argued

that a better formulation is to view the problem as one of Bayesian inference and rely on probabilistic image representations. In this paper we show that this formulation naturally leads to 1) representations with support for region-based interaction without segmentation and 2) intuitive mechanisms to account for both positive and negative user feedback.

## 2 Retrieval as Bayesian inference

The standard interaction paradigm for CBIR is the so-called "query by example", where the user provides the system with a few examples, and the system retrieves from the database images that are visually similar to these examples. The problem is naturally formulated as one of statistical classification: given a representation (or feature) space $\mathcal{F}$ the goal is to find a map $g : \mathcal{F} \to M = \{1, \ldots, K\}$ from $\mathcal{F}$ to the set $M$ of image classes in the database. $K$, the cardinality of $M$, can be as large as the number of items in the database (in which case each item is a class by itself), or smaller. If the goal of the retrieval system is to minimize the probability of error, it is well known that the optimal map is the Bayes classifier [3]

$$g^*(\mathbf{x}) = \arg\max_i P(S_i = 1|\mathbf{x}) = \arg\max_i P(\mathbf{x}|S_i = 1)P(S_i = 1) \tag{1}$$

where $\mathbf{x}$ are the example features provided by the user and $S_i$ is a binary variable indicating the selection of class $i$. In the absence of any prior information about which class is most suited for the query, an uninformative prior can be used and the optimal decision is the maximum likelihood criteria

$$g^*(\mathbf{x}) = \arg\max_i P(\mathbf{x}|S_i = 1). \tag{2}$$

Besides theoretical soundness, Bayesian retrieval has two distinguishing properties of practical relevance. First, because the features $\mathbf{x}$ in equation (1) can be any subset of a given query image, the retrieval criteria is valid for both region-based and image-based queries. Second, due to its probabilistic nature, the criteria also provides a basis for designing retrieval systems that can account for user-feedback through belief propagation.

## 3 Bayesian relevance feedback

Suppose that instead of a single query $\mathbf{x}$ we have a sequence of $t$ queries $\{\mathbf{x}_1, \ldots, \mathbf{x}_t\}$, where $t$ is a time stamp. By simple application of Bayes rule

$$P(S_i = 1|\mathbf{x}_1, \ldots, \mathbf{x}_t) = \gamma_t P(\mathbf{x}_t|S_i = 1)P(S_i = 1|\mathbf{x}_1, \ldots, \mathbf{x}_{t-1}), \tag{3}$$

where $\gamma_t$ is a normalizing constant and we have assumed that, given the knowledge of the correct image class, the current query $\mathbf{x}_t$ is independent of the previous ones. This basically means that the user provides the retrieval system with new information at each iteration of the interaction. Equation (3) is a simple but intuitive mechanism to integrate information over time. It states that the system's beliefs about the user's interests at time $t - 1$ simply become the prior beliefs for iteration $t$. New data provided by the user at time $t$ is then used to update these beliefs, which in turn become the priors for iteration $t + 1$. From a computational standpoint the procedure is very efficient since the only quantity that has to be computed at each time step is the likelihood of the data in the corresponding query. Notice that this is exactly equation (2) and would have to be computed even in the absence of any learning.

By taking logarithms and solving for the recursion, equation (3) can also be written as

$$\log P(S_i = 1|\mathbf{x}_1, \ldots, \mathbf{x}_t) = \sum_{k=0}^{t-1} \log \gamma_{t-k} + \sum_{k=0}^{t-1} \log P(\mathbf{x}_{t-k}|S_i = 1) + \log P(S_i = 1),$$

$$\tag{4}$$

exposing the main limitation of the belief propagation mechanism: for large $t$ the contribution, to the right-hand side of the equation, of the new data provided by the user is very small, and the posterior probabilities tend to remain constant. This can be avoided by penalizing older terms with a *decay factor* $\alpha_{t-k}$

$$
\log P(S_i = 1 | \mathbf{x}_1, \ldots, \mathbf{x}_t) = \sum_{k=0}^{t-1} \alpha_{t-k} \log \gamma_{t-k} + \sum_{k=0}^{t-1} \alpha_{t-k} \log P(\mathbf{x}_{t-k} | S_i = 1) + \\
\alpha_0 \log P(S_i = 1),
$$

where $\alpha_t$ is a monotonically decreasing sequence. In particular, if $\alpha_{t-k} = \alpha(1-\alpha)^k, \alpha \in (0, 1]$ we have

$$
\log P(S_i = 1 | \mathbf{x}_1, \ldots, \mathbf{x}_t) = \log \gamma_t' + \alpha \log P(\mathbf{x}_t | S_i = 1) + \\
(1-\alpha) \log P(S_i = 1 | \mathbf{x}_1, \ldots, \mathbf{x}_{t-1}).
$$

Because $\gamma_t'$ does not depend on $i$, the optimal class is

$$
S_i^* = \arg\max_i \{\alpha \log P(\mathbf{x}_t | S_i = 1) + (1-\alpha) \log P(S_i = 1 | \mathbf{x}_1, \ldots, \mathbf{x}_{t-1})\}. \tag{5}
$$

## 4 Negative feedback

In addition to positive feedback, there are many situations in CBIR where it is useful to rely on negative user-feedback. One example is the case of image classes characterized by overlapping densities. This is illustrated in Figure 1 a) where we have two classes with a common attribute (e.g. regions of blue sky) but different in other aspects (class A also contains regions of grass while class B contains regions of white snow). If the user starts with an image of class B (e.g. a picture of a snowy mountain), using regions of sky as positive examples is not likely to quickly take him/her to the images of class A. In fact, all other factors being equal, there is an equal likelihood that the retrieval system will return images from the two classes. On the other hand, if the user can explicitly indicate interest in regions of sky but not in regions of snow, the likelihood that only images from class A will be returned increases drastically.

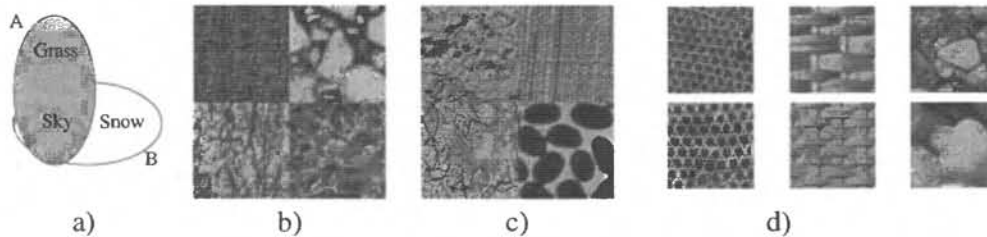

Figure 1: a) two overlapping image classes. b) and c) two images in the tile database. d) three examples of pairs of visually similar images that appear in different classes.

Another example of the importance of negative feedback are local minima of the search space. These happen when in response to user feedback, the system returns exactly the same images as in a previous iteration. Assuming that the user has already given the system all the possible positive feedback, the only way to escape from such minima is to choose some regions that are not desirable and use them as negative feedback. In the case of the example above, if the user gets stuck with a screen full of pictures of white mountains, he/she can simply select some regions of snow to escape the local minima.

In order to account for negative examples, we must penalize the classes under which these score well while favoring the classes that assign a high score to the positive examples.

Unlike positive examples, for which the likelihood is known, it is not straightforward to estimate the likelihood of a particular negative example given that the user is searching for a certain image class. We assume that the likelihood with which $\mathbf{y}$ will be used as a negative example given that the target is class $i$, is equal to the likelihood with which it will be used as a positive example given that the target is any other class. Denoting the use of $\mathbf{y}$ as a negative example by $\bar{\mathbf{y}}$, this can be written as

$$P(\bar{\mathbf{y}}|S_i = 1) = P(\mathbf{y}|S_i = 0). \qquad (6)$$

This assumption captures the intuition that a good negative example when searching for class $i$, is one that would be a good positive example if the user were looking for any class other than $i$. E.g. if class $i$ is the only one in the database that does not contain regions of sky, using pieces of sky as negative examples will quickly eliminate the other images in the database.

Under this assumption, negative examples can be incorporated into the learning by simply choosing the class $i$ that maximizes the posterior odds ratio [4] between the hypotheses "class $i$ is the target" and "class $i$ is not the target"

$$S_i^* = \arg\max_i \frac{P(S_i = 1|\mathbf{x}_t, \ldots, \mathbf{x}_1, \mathbf{y}_t, \ldots, \mathbf{y}_1)}{P(S_i = 0|\mathbf{x}_t, \ldots, \mathbf{x}_1, \mathbf{y}_t, \ldots, \mathbf{y}_1)} = \arg\max_i \frac{P(S_i = 1|\mathbf{x}_t, \ldots, \mathbf{x}_1)}{P(S_i = 0|\mathbf{y}_t, \ldots, \mathbf{y}_1)}$$

where $\mathbf{x}$ are the positive and $\bar{\mathbf{y}}$ the negative examples, and we have assumed that, given the positive (negative) examples, the posterior probability of a given class being (not being) the target is independent of the negative (positive) examples. Once again, the procedure of the previous section can be used to obtain a recursive version of this equation and include a decay factor which penalizes ancient terms

$$S_i^* = \arg\max_i \left\{ \alpha \log \frac{P(\mathbf{x}_t|S_i = 1)}{P(\mathbf{y}_t|S_i = 0)} + (1 - \alpha) \log \frac{P(S_i = 1|\mathbf{x}_1, \ldots, \mathbf{x}_{t-1})}{P(S_i = 0|\mathbf{y}_1, \ldots, \mathbf{y}_{t-1})} \right\}.$$

Using equations (4) and (6)

$$P(S_i = 0|\mathbf{y}_1, \ldots, \mathbf{y}_t) \quad \propto \quad \prod_k P(\mathbf{y}_k|S_i = 0) = \prod_k P(\bar{\mathbf{y}}_k|S_i = 1)$$

$$\propto \quad P(S_i = 1|\bar{\mathbf{y}}_1, \ldots, \bar{\mathbf{y}}_t),$$

we obtain

$$S_i^* = \arg\max_i \left\{ \alpha \log \frac{P(\mathbf{x}_t|S_i = 1)}{P(\bar{\mathbf{y}}_t|S_i = 1)} + (1 - \alpha) \log \frac{P(S_i = 1|\mathbf{x}_1, \ldots, \mathbf{x}_{t-1})}{P(S_i = 1|\bar{\mathbf{y}}_1, \ldots, \bar{\mathbf{y}}_{t-1})} \right\}. \qquad (7)$$

While maximizing the ratio of posterior probabilities is a natural way to favor image classes that explain well the positive examples and poorly the negative ones, it tends to over-emphasize the importance of negative examples. In particular, any class with zero probability of generating the negative examples will lead to a ratio of $\infty$, even if it explains very poorly the positive examples. To avoid this problem we proceed in two steps:

- start by solving equation (5), i.e. sort the classes according to how well they explain the positive examples.
- select the subset of the best $N$ classes and solve equation (7) considering only the classes in this subset.

## 5 Experimental evaluation

We performed experiments to evaluate 1) the accuracy of Bayesian retrieval on region-based queries and 2) the improvement in retrieval performance achievable with relevance

feedback. Because in a normal browsing scenario it is difficult to know the ground truth for the retrieval operation (at least without going through the tedious process of hand-labeling all images in the database), we relied instead on a controlled experimental set up for which ground truth is available. All experiments reported on this section are based on the widely used Brodatz texture database which contains images of 112 textures, each of them being represented by 9 different patches, in a total of 1008 images. These were split into two groups, a small one with 112 images (one example of each texture), and a larger one with the remaining 896. We call the first group the *test* database and the second the *Brodatz* database. A synthetic database with 2000 images was then created from the larger set by randomly selecting 4 images at a time and making a $2 \times 2$ tile out of them. Figure 1 b) and c) are two examples of these tiles. We call this set the *tile* database.

## 5.1 Region-based queries

We performed two sets of experiments to evaluate the performance of region-based queries. In both cases the test database was used as a test set and the image features were the coefficients of the discrete cosine transform (DCT) of an $8 \times 8$ block-wise image decomposition over a grid containing every other image pixel. The first set of experiments was performed on the Brodatz database while the tile database was used in the second. A mixture of 16 Gaussians was estimated, using EM, for each of the images in the two databases.

In both sets of experiments, each query consisted of selecting a few image blocks from an image in the test set, evaluating equation (2) for each of the classes and returning those that best explained the query. Performance was measured in terms of precision (percent of the retrieved images that are relevant to the query) and recall (percent of the relevant images that are retrieved) averaged over the entire test set. The query images contained a total of 256 non-overlapping blocks. The number of these that were used in each query varied between 1 (0.3 %) of the image size) and 256 (100 %). Figure 2 depicts precision-recall plots as a function of this number.

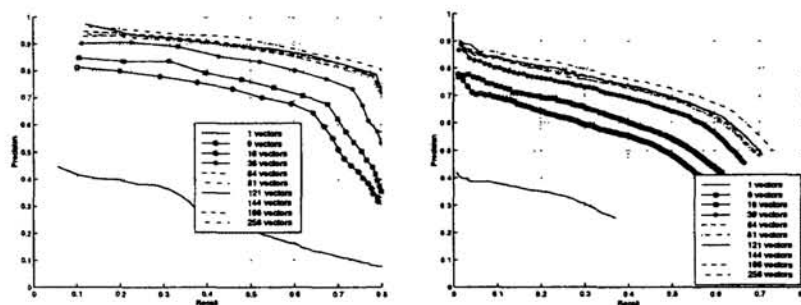

Figure 2: Precision-recall curves as a function of the number of feature vectors included in the query. Left: Brodatz database. Right: Tile database.

The graph on the left is relative to the Brodatz database. Notice that precision is generally high even for large values of recall and the performance increases quickly with the percentage of feature vectors included in the query. In particular, 25% of the texture patch (64 blocks) is enough to achieve results very close to those obtained with all pixels. This shows that the retrieval criteria is robust to missing data. The graph on the left presents similar results for the tile database. While there is some loss in performance, this loss is not dramatic - a decrease between 10 and 15 % in precision for any given recall. In fact, the results are still good: when a reasonable number of feature vectors is included in the query, about 8.5 out of the 10 top retrieved images are, on average, relevant. Once again, performance improves rapidly with the number of feature vectors in the query and 25% of

the image is enough for results comparable to the best. This confirms the argument that Bayesian retrieval leads to effective region-based queries even for imagery composed by multiple visual stimulae.

## 5.2 Learning

The performance of the learning algorithm was evaluated on the tile database. The goal was to determine if it is possible to reach a desired target image by starting from a weakly related one and providing positive and negative feedback to the retrieval system. This simulates the interaction between a real user and the CBIR system and is an iterative process, where each iteration consists of selecting a few examples, using them as queries for retrieval and examining the top $M$ retrieved images to find examples for the next iteration. $M$ should be small since most users are not willing to go through lots of false positives to find the next query. In all experiments we set $M = 10$ corresponding to one screenful of images.

The most complex problem in testing is to determine a good strategy for selecting the examples to be given to the system. The closer this strategy is to what a real user would do, the higher the practical significance of the results. However, even when there is clear ground truth for the retrieval (as is the case of the tile database) it is not completely clear how to make the selection. While it is obvious that regions of texture classes that appear in the target should be used as positive feedback, it is much harder to determine automatically what are good negative examples. As Figure 1 d) illustrates, there are cases in which textures from two different classes are visually similar. Selecting images from one of these classes as a negative example for the other will be a disservice to the learner.

While real users tend not to do this, it is hard to avoid such mistakes in an automatic setting, unless one does some sort of pre-classification of the database. Because we wanted to avoid such pre-classification we decided to stick with a simple selection procedure and live with these mistakes. At each step of the iteration, examples were selected in the following way: among the 10 top images returned by the retrieval system, the one with most patches from texture classes also present in the target image was selected to be the next query. One block from each patch in the query was then used as a positive (negative) example if the class of that patch was also (was not) represented in the target image.

This strategy is a worst-case scenario. First, the learner might be confused by conflicting negative examples. Second, as seen above, better retrieval performance can be achieved if more than one block from each region is included in the queries. However, using only one block reduced the computational complexity of each iteration, allowing us to average results over several runs of the learning process. We performed 100 runs with randomly selected target images. In all cases, the initial query image was the first in the database containing one class in common with the target.

The performance of the learning algorithm can be evaluated in various ways. We considered two metrics: the percentage of the runs which converged to the right target, and the number of iterations required for convergence. Because, to prevent the learner from entering loops, any given image could only be used once as a query, the algorithm can diverge in two ways. Strong divergence occurs when, at a given time step, the images (among the top 10) that can be used as queries do not contain any texture class in common with the target. In such situation, a real user will tend to feel that the retrieval system is incoherent and abort the search. Weak divergence occurs when all the top 10 images have previously been used. This is a less troublesome situation because the user could simply look up more images (e.g. the next 10) to get new examples.

We start by analyzing the results obtained with positive feedback only. Figure 3 a) and b) present plots of the convergence rate and median number of iterations as a function of the decay factor $\alpha$. While when there is no learning ($\alpha = 1$) only 43% of the runs converge,

the convergence rate is always higher when learning takes place and for a significant range of $\alpha$ ($\alpha \in [0.5, 0.8]$) it is above 60%. This not only confirms that learning can lead to significant improvements of retrieval performance but also shows that a precise selection of $\alpha$ is not crucial. Furthermore, when convergence occurs it is usually very fast, taking from 4 to 6 iterations. On the other hand, a significant percentage of runs do not converge and the majority of these are cases of strong divergence.

As illustrated by Figure 3 c) and d), this percentage decreases significantly when both positive and negative examples are allowed. The rate of convergence is in this case usually between 80 and 90 % and strong divergence never occurs. And while the number of iterations for convergence increases, convergence is still fast (usually below 10 iterations). This is indeed the great advantage of negative examples: they encourage some exploration of the database which avoids local minima and leads to convergence. Notice that, when there is no learning, the convergence rate is high and learning can actually increase the rate of divergence. We believe that this is due to the inconsistencies associated with the negative example selection strategy. However, when convergence occurs, it is always faster if learning is employed.

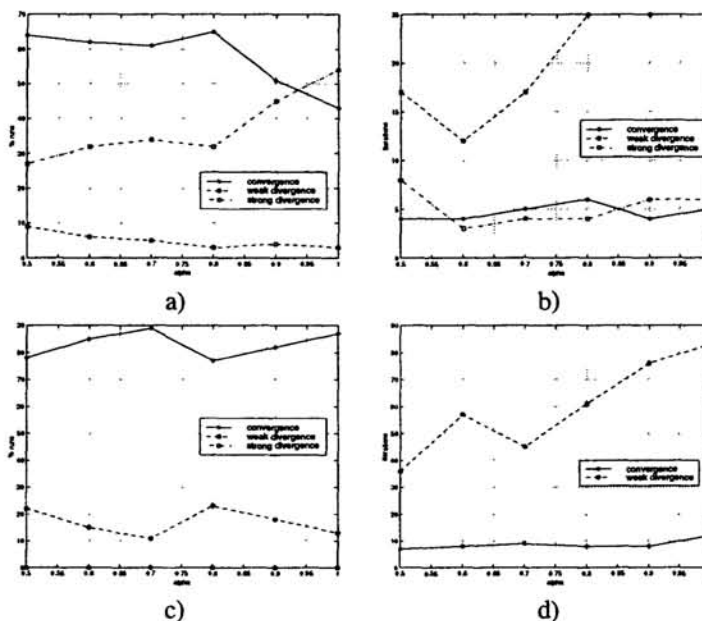

a)  b)

c)  d)

Figure 3: Learning performance as a function of $\alpha$. Left: Percent of runs which converged. Right: Median number of iterations. Top: positive examples. Bottom: positive and negative examples.

# References

[1] S. Belongie, C. Carson, H. Greenspan, and J. Malik. Color-and texture-based image segmentation using EM and its application to content-based image retrieval. In *International Conference on Computer Vision*, pages 675–682, Bombay, India, 1998.

[2] I. Cox, M. Miller, S. Omohundro, and P. Yianilos. PicHunter: Bayesian Relevance Feedback for Image Retrieval. In *Int. Conf. on Pattern Recognition*, Vienna, Austria, 1996.

[3] L. Devroye, L. Gyorfi, and G. Lugosi. *A Probabilistic Theory of Pattern Recognition*. Springer-Verlag, 1996.

[4] A. Gelman, J. Carlin, H. Stern, and D. Rubin. *Bayesian Data Analysis*. Chapman Hall, 1995.

[5] A. Pentland, R. Picard, and S. Sclaroff. Photobook: Content-based Manipulation of Image Databases. *International Journal of Computer Vision*, Vol. 18(3):233–254, June 1996.
